# Simplifying Mixture Models
# through Function Approximation

**Kai Zhang     James T. Kwok**
Department of Computer Science and Engineering
The Hong Kong University of Science and Technology
Clear Water Bay, Kowloon, Hong Kong
{twinsen, jamesk}@cse.ust.hk

## Abstract

Finite mixture model is a powerful tool in many statistical learning problems. In this paper, we propose a general, structure-preserving approach to reduce its model complexity, which can bring significant computational benefits in many applications. The basic idea is to group the original mixture components into compact clusters, and then minimize an upper bound on the approximation error between the original and simplified models. By adopting the $L_2$ norm as the distance measure between mixture models, we can derive closed-form solutions that are more robust and reliable than using the KL-based distance measure. Moreover, the complexity of our algorithm is only linear in the sample size and dimensionality. Experiments on density estimation and clustering-based image segmentation demonstrate its outstanding performance in terms of both speed and accuracy.

## 1  Introduction

In many statistical learning problems, it is useful to obtain an estimate of the underlying probability density given a set of observations. Such a density model can facilitate discovery of the underlying data structure in unsupervised learning, and can also yield, asymptotically, optimal discriminant procedures [7]. In this paper, we focus on the *finite mixture model*, which describes the distribution by a mixture of simple parametric functions $\phi(\cdot)$'s as $f(\mathbf{x}) = \sum_{j=1}^{n} \alpha_j \phi(\mathbf{x}, \theta_j)$. Here, $\theta_j$ is the parameter for the $j$th component, and the mixing parameters $\alpha_j$'s satisfy $\sum_{j=1}^{n} \alpha_j = 1$. The most common parametric form of $\phi$ is the Gaussian, leading to the well-known *Gaussian mixtures*.

The mixture model has been widely used in clustering and density estimation, where the model parameters can be estimated by the standard Expectation-Maximization (EM) algorithm. However, the EM can be prohibitively expensive on large problems [12]. On the other hand, note that in many learning processes using mixture models (such as particle filtering [6] and non-parametric belief propagation [13]), the computational requirement is also very demanding due to the large number of components involved in the model. In this situation, our interest is more on reducing the number of components for prospective computational saving. Previous works typically employ spatial data structures, such as the kd-tree [8, 9], for acceleration. Recently, [5] proposes to reduce a large Gaussian mixture into a smaller one by minimizing a KL-based distance between the two mixtures. This has been applied with success on hierarchical clustering of scenery images and handwritten digits.

In this paper, we propose a new algorithm for simplifying a given finite mixture model while preserving its component structures, with application to nonparametric density estimation and clustering. The idea is to minimize an upper bound on the approximation error between the original and simplified mixture models. By adopting the $L_2$ norm as the error criterion, we can derive closed-form solutions that are more robust and reliable than using the KL-based distance measures. At the same

time, our algorithm can be applied to general Gaussian kernels, and the complexity is only linear in the sample size and dimensionality.

The rest of the paper is organized as follows. In Section 2 we describe the proposed approach in detail, and illustrate its advantages compared with the existing ones. In Section 3, we report experimental results on simplifying the Parzen window estimator, and color image segmentation through the mean shift clustering procedure. Section 4 gives some concluding remarks.

## 2   Approximation Algorithm

Given a mixture model

$$f(\mathbf{x}) = \sum_{j=1}^{n} \alpha_j \phi_j(\mathbf{x}), \tag{1}$$

we assume that the $j$th component $\phi_j(\mathbf{x})$ is of the form

$$\phi_j(\mathbf{x}) = |\mathbf{H}_j|^{-1/2} K_{\mathbf{H}_j}(\mathbf{x} - \mathbf{x}_j), \tag{2}$$

with weight $\alpha_j$, center $\mathbf{x}_j$ and covariance matrix $\mathbf{H}_j$. Here, $K_{\mathbf{H}}(\mathbf{x}) = K(\mathbf{H}^{-1/2}\mathbf{x})$ where $K(\mathbf{x})$ is the kernel that is bounded and has compact support. Note that for radially symmetric kernels, it suffices to define $K$ by the *profile* $k$ such that $K(\mathbf{x}) = k(\|\mathbf{x}\|^2)$. With this notion, the gradient of the kernel function, $K_{\mathbf{H}}(\mathbf{x})$, can be conveniently written as $\partial_{\mathbf{x}} K_{\mathbf{H}}(\mathbf{x}) = k'(r)\partial_{\mathbf{x}} r = 2k'(r)\mathbf{H}^{-1}\mathbf{x}$, where $r = \mathbf{x}\mathbf{H}^{-1}\mathbf{x}$. Our task is to approximate $f$ with a simpler mixture model

$$g(\mathbf{x}) = \sum_{i=1}^{m} w_i g_i(\mathbf{x}), \tag{3}$$

with $m \ll n$, where each component $g_i$ also takes the form

$$g_i(\mathbf{x}) = |\tilde{\mathbf{H}}_i|^{-1/2} K_{\tilde{\mathbf{H}}_i}(\mathbf{x} - \mathbf{t}_i), \tag{4}$$

with weight $w_i$, center $\mathbf{t}_i$, and covariance matrix $\tilde{\mathbf{H}}_i$.

Note that direct approximation of $f$ by $g$ is not feasible, because they involve a large number of components. Given a distance measure $D(\cdot, \cdot)$ between functions, the approximation error

$$\mathcal{E} = D(f, g) = D\left(\sum_{j=1}^{n} \alpha_j \phi_j, \sum_{i=1}^{m} w_i g_i\right) \tag{5}$$

is usually difficult to optimize. However, the problem can be very much simplified by minimizing an upper bound of $\mathcal{E}$. Consider the $L_2$ distance $D(\phi, \phi') = \int (\phi(\mathbf{x}) - \phi'(\mathbf{x}))^2 \, d\mathbf{x}$, and suppose that the mixture components $\{\phi_j\}_{j=1}^{n}$ are divided into disjoint clusters $S_1, \ldots, S_m$. Then, it is easy to see that the approximation error $\mathcal{E}$ is bounded by

$$\mathcal{E} = \int \left(\sum_{j=1}^{n} \alpha_j \phi_j(\mathbf{x}) - \sum_{i=1}^{m} w_i g_i(\mathbf{x})\right)^2 d\mathbf{x} \le m \sum_{i=1}^{m} \int \left(w_i g_i(\mathbf{x}) - \sum_{j \in S_i} \alpha_j \phi_j(\mathbf{x})\right)^2 d\mathbf{x}.$$

Denote this upper bound by $\overline{\mathcal{E}} = m \sum_{i=1}^{m} \overline{\mathcal{E}}_i$, where

$$\overline{\mathcal{E}}_i = \int \left(w_i g_i(\mathbf{x}) - \sum_{j \in S_i} \alpha_j \phi_j(\mathbf{x})\right)^2 d\mathbf{x}. \tag{6}$$

Note that $\overline{\mathcal{E}}$ is the sum of the "local" approximation errors $\overline{\mathcal{E}}_i$'s. Hence, if we can find a good representative $w_i g_i$ for each cluster by minimizing the local approximation error $\overline{\mathcal{E}}_i$, the overall approximation performance can also be guaranteed. This suggests partitioning the original mixture components into compact clusters, wherein approximation can then be done much more easily. Our basic algorithm proceeds as follows:

*1.* (Section 2.1.1) Partition the set of mixture components ($\phi_j$'s) into $m$ clusters where $m \ll n$. Let $S_i$ be the set that indexes all components belonging to the $i$th cluster.

*2.* (Section 2.1.2) For each cluster, approximate the local mixture model $\sum_{j \in S_i} \alpha_j \phi_j$ by a single component $w_i g_i$, where $g_i$ is defined in (4).

*3.* The simplified model $g$ is obtained by $g(\mathbf{x}) = \sum_{i=1}^{m} w_i g_i(\mathbf{x})$.

These steps will be discussed in more detail in the following sections.

## 2.1 Procedure

### 2.1.1 Partitioning of Components

In this section, we consider how to group similar components into the same cluster, so that the subsequent local approximation can be more accurate. A useful algorithm for this task is the classic *vector quantization* (VQ) [4], where one iterates between partitioning a set of vectors and finding the best prototype for each partition until the distortion error converges. By defining a distance $D(\cdot, \cdot)$ between mixture components $\phi_j$'s, we can partition the mixture components in a similar way. However, vector quantization is sensitive to the initial partitioning. So we first introduce a simple but highly efficient partitioning method called *sequential sampling* (SS):

*1.* Randomly select a $\phi_j$ and add it to the set of representatives $\mathcal{R}$.

*2.* For all the components ($j = 1, 2, \ldots, n$), do the following
- Compute the distance $D(\phi_j, \mathcal{R}_i)$, where $\mathcal{R}_i \in \mathcal{R}$.
- Once if $D(\phi_j, \mathcal{R}_i) \leq r$, where $r$ is a predefined threshold, assign $\phi_j$ to the representative $\mathcal{R}_i$, and then process the next component.
- If $D(\phi_j, \mathcal{R}_i) > r$ for all $\mathcal{R}_i \in \mathcal{R}$, add $\phi_j$ as a new representative of $\mathcal{R}$.

*3.* Terminate when all the components have been processed.

This procedure partitions the components by choosing those $\phi_j$'s that are enough far away as representatives, with a user-defined resolution $r$. So it is less sensitive to initialization. In practice, we will first initialize by sequential sampling, and then perform the iterative VQ procedure to further refine the partition, i.e., find the best representative $\mathcal{R}_i$ for each cluster, reassign each component $\phi_j$ to the closest representative $\mathcal{R}_{\pi(j)}$, and iterate until the error $\sum_j \alpha_j D(\phi_j, \mathcal{R}_{\pi(j)})$ converges.

### 2.1.2 Local Approximation

In this part, we consider how to obtain a good representative, $w_i g_i$ in (4), for each local cluster $S_i$. The task is to determine the unknown variables $w_i, \mathbf{t}_i$ and $\tilde{\mathbf{H}}_i$ associated with $g_i$. Using the $L_2$ norm, the upper bound (6) of the local approximation error can be written as

$$
\begin{aligned}
\overline{\mathcal{E}}_i &= \int \left( w_i g_i(\mathbf{x}) - \sum_{j \in S_i} \alpha_j \phi_j(\mathbf{x}) \right)^2 d\mathbf{x} \\
&= w_i^2 \frac{C_K}{|2\tilde{\mathbf{H}}_i|^{1/2}} - w_i \sum_{j \in S_i} \frac{2 C_K \alpha_j k(r_{ij})}{|\mathbf{H}_j + \tilde{\mathbf{H}}_i|^{1/2}} + c_i.
\end{aligned}
$$

Here, $C_K = \int k(\mathbf{x}'\mathbf{x}) d\mathbf{x}$ is a kernel-dependent constant, $c_i = \int (\sum_{j \in S_i} \alpha_j \phi_j^2(\mathbf{x}))^2 d\mathbf{x}$ is a data-dependent constant (irrelevant to the unknown variables), and $r_{ij} = (\mathbf{t}_i - \mathbf{x}_j)'(\mathbf{H}_j + \tilde{\mathbf{H}}_i)^{-1}(\mathbf{t}_i - \mathbf{x}_j)$. Here we have assumed that $k(a) \cdot k(b) = k(a + b)$, which is valid for the Gaussian and negative exponential kernels. Without this assumption, solutions can still be obtained but are less compact.

To minimize $\overline{\mathcal{E}}_i$ w.r.t. $w_i, \mathbf{t}_i$ and $\tilde{\mathbf{H}}_i$, one can set the corresponding partial derivatives of $\overline{\mathcal{E}}_i$ to zero. However, this leads to a nonlinear system that is quite difficult to solve. Here, we decouple the relations among these three parameters. First, observe that $\overline{\mathcal{E}}_i$ is a quadratic function of $w_i$. Therefore, given $\tilde{\mathbf{H}}_i$ and $\mathbf{t}_i$, the minimum value of $\overline{\mathcal{E}}_i$ can be easily obtained as

$$
\overline{\mathcal{E}}_i^{\min} = |\tilde{\mathbf{H}}_i|^{\frac{1}{2}} \left( \sum_{j \in S_i} \alpha_j k(r_{ij}) \left| \mathbf{H}_j + \tilde{\mathbf{H}}_i \right|^{-1/2} \right)^2. \tag{7}
$$

The remaining task is to minimize $\overline{\mathcal{E}}_i^{\min}$ w.r.t. $\mathbf{t}_i$ and $\tilde{\mathbf{H}}_i$. By setting $\partial_{\mathbf{t}_i} \overline{\mathcal{E}}_i^{min} = 0$, we have

$$\mathbf{t}_i = \mathbf{M}_i^{-1} \sum_{j \in S_i} \frac{\alpha_j k'\left(r_{ij}\right)\left(\mathbf{H}_j + \tilde{\mathbf{H}}_i\right)^{-1} \mathbf{x}_j}{|\mathbf{H}_j + \tilde{\mathbf{H}}_i|^{1/2}}, \tag{8}$$

where

$$\mathbf{M}_i = \sum_{j \in S_i} \frac{\alpha_j k'\left(r_{ij}\right)\left(\mathbf{H}_j + \tilde{\mathbf{H}}_i\right)^{-1}}{|\mathbf{H}_j + \tilde{\mathbf{H}}_i|^{1/2}}.$$

This is an iterative contraction mapping. If $\tilde{\mathbf{H}}_i$ is fixed, we can obtain $\mathbf{t}_i$ by starting with an initial $\mathbf{t}_i^{(0)}$, and then iterate (8) until convergence. Now, to solve $\tilde{\mathbf{H}}_i$, we set $\partial_{\tilde{\mathbf{H}}_i} \overline{\mathcal{E}}_i^{min} = 0$ and obtain

$$\tilde{\mathbf{H}}_i = \mathbf{P}_i^{-1} \sum_{j \in S_i} \frac{\alpha_j (\tilde{\mathbf{H}}_i + \mathbf{H}_j)^{-1}}{|\mathbf{H}_j + \tilde{\mathbf{H}}_i|^{1/2}} \left( k(r_{ij})\mathbf{H}_j + 4(-k'(r_{ij}))(\mathbf{x}_j - \mathbf{t}_i)(\mathbf{x}_j - \mathbf{t}_i)'(\tilde{\mathbf{H}}_i + \mathbf{H}_j)^{-1}\tilde{\mathbf{H}}_i \right), \tag{9}$$

where

$$\mathbf{P}_i = \sum_{j \in S_i} \frac{(\tilde{\mathbf{H}}_i + \mathbf{H}_j)^{-1}}{|\mathbf{H}_j + \tilde{\mathbf{H}}_i|^{1/2}} \alpha_j k(r_{ij}).$$

In summary, we first initialize

$$\begin{aligned}
\mathbf{t}_i^{(0)} &= \textstyle\sum_{j \in S_i} \alpha_j \mathbf{x}_j / (\sum_{j \in S_i} \alpha_j), \\
\tilde{\mathbf{H}}_i^{(0)} &= \textstyle\sum_{j \in S_i} \alpha_j \left( \mathbf{H}_j + (\mathbf{t}_i^{(0)} - \mathbf{x}_j)(\mathbf{t}_i^{(0)} - \mathbf{x}_j)' \right) / (\sum_{j \in S_i} \alpha_j),
\end{aligned}$$

and then iterate (8) and (9) until convergence. The converged values of $\mathbf{t}_i$ and $\tilde{\mathbf{H}}_i$ are substituted into $\partial_{w_i} \overline{\mathcal{E}}_i = 0$ to obtain $w_i$ as

$$w_i = |2\tilde{\mathbf{H}}_i|^{\frac{1}{2}} \sum_{j \in S_i} \frac{\alpha_j k(r_{ij})}{|\mathbf{H}_j + \tilde{\mathbf{H}}_i|^{1/2}}. \tag{10}$$

## 2.2 Complexity

In the partitioning step, sequential sampling has a complexity of $O(dmn)$, where $n$ is original model size, $m$ is the number of clusters, and $d$ the dimension. By using a hierarchical scheme [2], this can be reduced to $O(dn \log(m))$. The VQ takes $O(dnm)$ time. In the local approximation step, the complexity is $l \sum_{i=1}^{m} n_i d^3 = lnd^3$, where $l$ is the maximum number of iterations needed. In practice, we can enforce a diagonal structure on the covariance matrix $\tilde{\mathbf{H}}_i$'s while still obtaining a closed-form solution. Hence, the complexity becomes linear in the dimension $d$ instead of cubic. Summing up these three terms, the overall complexity is $O(dn \log(m) + dnm + lnd) = O(dn(m + l))$, which is linear in both the data size and dimension (in practice $m$ and $l$ are quite small).

## 2.3 Remarks

In this section, we discuss some interesting properties of the approximation scheme proposed in Section 2.1.2. To have better intuitions, we examine the special case of a Parzen window density estimator [11], where all $\phi_j$'s have the same weights and bandwidths ($\mathbf{H}_j = \mathbf{H}$ for $j = 1, 2, \ldots, n$). Equation (9) then reduces to

$$\tilde{\mathbf{H}}_i = \mathbf{H} + 4\tilde{\mathbf{H}}_i(\tilde{\mathbf{H}}_i + \mathbf{H})^{-1}\mathbf{V}_i, \tag{11}$$

where

$$\mathbf{V}_i = \frac{\sum_{j \in S_i} \alpha_j(-k'(r_{ij}))(\mathbf{x}_j - \mathbf{t}_i)(\mathbf{x}_j - \mathbf{t}_i)'}{\sum_{j \in S_i} \alpha_j k(r_{ij})}.$$

It shows that the bandwidth $\tilde{\mathbf{H}}_i$ of $g_i$ can be decomposed into two parts: the bandwidth $\mathbf{H}$ of the original kernel density estimator, and the covariance $\mathcal{V}_i$ of the local cluster $S_i$ with an adjusting matrix $\Gamma_i = 4\tilde{\mathbf{H}}_i(\tilde{\mathbf{H}}_i + \mathbf{H})^{-1}$. As an illustration, consider the 1-D case where $\mathbf{H} = h^2$, $\tilde{\mathbf{H}}_i = h_i^2$.

Then $\gamma_i = \frac{4h_i^2}{h^2+h_i^2}$, and $h_i^2 = h^2 + \gamma_i \mathcal{V}_i$. Since $\mathcal{V}_i \geq 0$ and $\gamma_i \geq 2$, we can see that $h_i^2 \geq h^2 + \mathcal{V}_i$. Moreover, $h_i$ is closely related to the spread of the local cluster. If all the points in $S_i$ are located at the same position (i.e., $\mathcal{V}_i = 0$), then $h_i^2 = h^2$. Otherwise, the larger the spread of the local cluster, the larger is $h_i$. In other words, the bandwidths $\tilde{\mathbf{H}}_i$'s are adaptive to the local data distribution.

Related works in simplifying the mixture models (such as [5]) simply choose $\tilde{\mathbf{H}}_i = \mathbf{H} + Cov[S_i]$. In comparison, our covariance term $\mathcal{V}_i$ is more reliable in that it incorporates distance-based weighting. Interestingly, this is somewhat similar to the bandwidth matrix used in the manifold Parzen windows [14], which is designed for handling sparse, high-dimensional data more robustly. Note that our choice of $\tilde{\mathbf{H}}_i$ is derived rigorously by minimizing the $L_2$ approximation error. Therefore, this coincidence naturally indicates the robustness of the $L_2$-norm based distance measures. Moreover, note that the adjusting matrix $\Gamma_i$ changes not only the scale of the bandwidth, but also its eigen-structures in an iterative manner. This will be very beneficial in multivariate cases.

Second, in determining the center of $g_i$, (8) can be reduced to

$$\mathbf{t}_i = \frac{\sum_{j \in S_i} \alpha_j k'_{\mathbf{H}+\tilde{\mathbf{H}}_i}(\mathbf{x}_j - \mathbf{t}_i)\,\mathbf{x}_j}{\sum_{j \in S_i} \alpha_j k'_{\mathbf{H}+\tilde{\mathbf{H}}_i}(\mathbf{x}_j - \mathbf{t}_i)}. \tag{12}$$

This can be regarded a mean-shift procedure [1] in the $d$-dimensional space with kernel $K$. It is easy to verify that this iterative procedure is indeed locating the peak of the density function $p_i(\mathbf{x}) = |\mathbf{H} + \tilde{\mathbf{H}}_i|^{-\frac{1}{2}} \sum_{j \in S_i} K_{\mathbf{H}+\tilde{\mathbf{H}}_i}(\mathbf{x} - \mathbf{x}_j)$. Note, on the other hand, that what we want to approximate originally is the local density $f_i(\mathbf{x}) = |\mathbf{H}|^{-\frac{1}{2}} \sum_{j \in S_i} K_{\mathbf{H}}(\mathbf{x} - \mathbf{x}_j)$. In the 1-D case (with $\mathbf{H} = h^2$, and $\tilde{\mathbf{H}}_i = h_i^2$), the bandwidth of $p_i$ (i.e., $h^2 + h_i^2$) is larger than that of $f_i$ (i.e., $h^2$).

It appears intriguing that on fitting a kernel density $f_i(\mathbf{x})$ estimated on the sample set $\{\mathbf{x}_j\}_{j \in S_i}$, one needs to locate the maximum of another density function $p_i(\mathbf{x})$, instead of the maximum of $f_i(\mathbf{x})$ itself or simply, the mean of the sample set $\{\mathbf{x}_j\}_{j \in S_i}$ as chosen in [5]. Indeed, these three choices coincide when the distribution of $S_i$ is symmetric and uni-modal, but will differ otherwise. Intuitively, when the data is asymmetric, the center $\mathbf{t}_i$ should be biased towards the heavier side of the data distribution. The maximum of $f_i(\mathbf{x})$ thus fails to meet this requirement. On the other hand, the mean of $S_i$, though biased towards the heavier side, still lacks an accurate control on the degree of bias. In comparison, our method provides a principled way of selecting the center. Note that $p_i(\mathbf{x})$ has a larger bandwidth than the original $f_i(\mathbf{x})$. Therefore, its maximum will move towards the heavier side of the distribution compared with that of $f_i(\mathbf{x})$, with the degree of bias automatically controlled by the mean shift iterations in (12).

Here, we give an illustration on the performance of the three center selection schemes. Figure 1(a) shows the histogram of a local cluster $S_i$, whose Parzen window estimator ($f_i$) is asymmetric. Figure 1(b) plots the corresponding approximation error $\overline{\mathcal{E}}_i$ (6) at different bandwidths $h_i$ (the remaining parameter, $w_i$, is set to the optimal value by (10) ). As can be seen, the approximation error of our method is consistently lower than those of the other two. Moreover, the resultant optimum is also much lower.

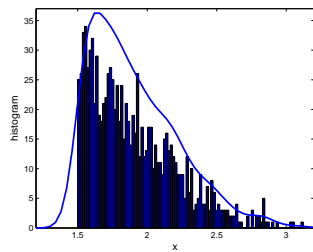

(a) The histogram of a local cluster $S_i$ and its density $f_i$.

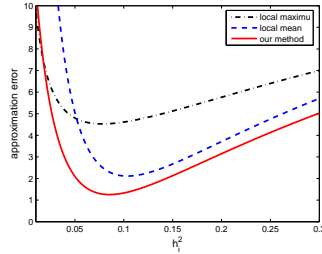

(b) Approximation error.

Figure 1: Approximation of an asymmetric density using different center selection schemes.

# 3 Experiments

In this section, we perform experiments to evaluate the performance of our mixture simplification scheme. We focus on the Parzen window estimator which, on given a set of samples $S = \{\mathbf{x}_i\}_{i=1}^n$ in $\mathbb{R}^d$, can be written as $\hat{f}(\mathbf{x}) = \frac{1}{n}|\mathbf{H}|^{-\frac{1}{2}} \sum_{j=1}^n K_{\mathbf{H}}(\mathbf{x} - \mathbf{x}_j)$. Note that the Parzen window estimator is a limiting form of the mixture model, where the number of components equals the data size and can be quite huge. In Section 3.1, we use the proposed approach to reduce the number of components in the kernel density estimator, and compare its performance with the algorithm in [5]. Then, in Section 3.2, we perform color image segmentation by running the mean shift clustering algorithm on the simplified density model.

## 3.1 Simplifying Nonparametric Density Models

In this section, we reduce the number of kernels in the Parzen window estimator by using the proposed approach and the method in [5]. Experiments are performed on a 1-D set with 1800 samples drawn from the Gaussian mixture $\frac{8}{18}\mathcal{N}(-2.6, 0.09) + \frac{6}{18}\mathcal{N}(-0.8, 0.36) + \frac{4}{18}\mathcal{N}(1.7, 0.64)$, where $\mathcal{N}(\mu, \sigma^2)$ denotes the normal distribution with mean $\mu$ and variance $\sigma^2$. The Gaussian kernel with fixed bandwidth $h = 0.3$ is used for density estimation. To make the problem more challenging, we choose $m = 5$, i.e., only 5 kernels are used to approximate the density. The $k$-means algorithm is used for initialization. As can be seen from Figure 2(b), the third Gaussian component has been broken into two by the method in [5]. In comparison, our result in Figure 2(c) is more reliable.

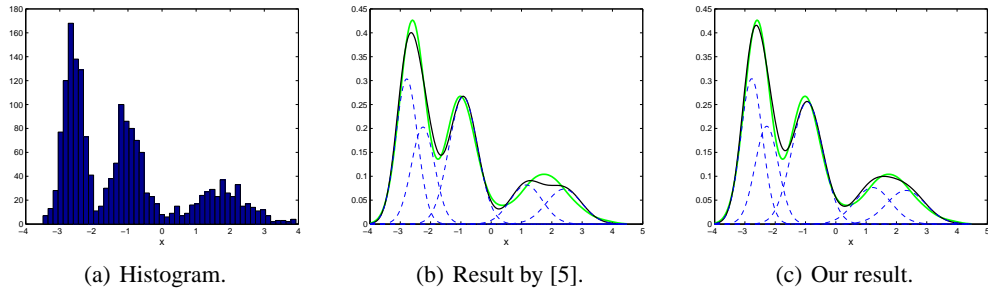

| (a) Histogram. | (b) Result by [5]. | (c) Our result. |

Figure 2: Approximate the Parzen window estimator by simplifying mixture models. Green: Parzen window estimator; black: simplified mixture model; blue-dashed: components of the mixture model.

To have a quantitative evaluation, we randomly generate the 3-Gaussian data 100 times, and compare the two algorithms (ours and [5]) using the following error criteria: 1) the $L_2$ error (5); 2) the standard KL distance; 3) the local KL-distance used in [5]. The local KL-distance between two mixtures, $f = \sum_{j=1}^n \alpha_j \phi_j$ and $g = \sum_{i=1}^m w_i g_i$, is defined as

$$d(f, g) = \sum_{j=1}^n \alpha_j KL(\phi_j || g_{\pi(j)}),$$

where $\pi(j)$ is the function that maps each component $\phi_j$ to the closest representative component $g_{\pi(j)}$ such that $\pi(j) = \arg\min_{i=1,2,\dots,m} KL(\phi_j || g_i)$.

Results are plotted in Figure 3, where for clarity we order the results in increasing error obtained by [5]. We can see that under the $L_2$ norm, the error of our algorithm is significantly lower than that of [5]. Quantitatively, our error is only about $36.61\%$ of that by [5]. On using the standard KL-distance, our error is about $87.34\%$ of that by [5], where the improvement is less significant. This is because the KL-distance is sensitive to the tail of the distribution, i.e., a small difference in the low-density regions may induce a huge KL-distance. As for the local KL-distance, our error is about $99.35\%$ of that by [5].

## 3.2 Image Segmentation

The Parzen window estimator can be used to reveal important clustering information, namely that its modes (or local maxima) correspond to dominant clusters in the data. This property is utilized in the

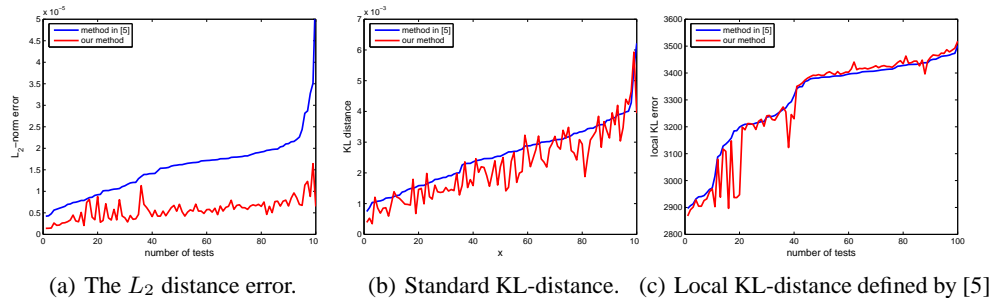

(a) The $L_2$ distance error.    (b) Standard KL-distance.    (c) Local KL-distance defined by [5]

Figure 3: Quantitative comparison of the approximation errors.

mean shift clustering algorithm [1, 3], where every data point is moved along the density gradient until it reaches the nearest local density maximum. The mean shift algorithm is robust, and can identify arbitrarily-shaped clusters in the feature space.

Recently, mean shift is applied in color image segmentation and has proven to be quite successful [1]. The idea is to identify homogeneous image regions through clustering in a properly selected feature space (such as color, texture, or shape). However, mean shift can be quite expensive due to the large number of kernels involved in the density estimator. To reduce the computational requirement, we first reduce the density estimator $\hat{f}(\mathbf{x})$ to a simpler model $g(\mathbf{x})$ using our simplification scheme, and then apply the iterative mean shift procedure on the simplified model $g(\mathbf{x})$.

Experiments are performed on a number of benchmark images[1] used in [1]. We use the Gaussian kernel with bandwidth $h = 20$. The partition parameter is $r = 25$. For comparison, we also implement the standard mean shift and its fast version using kd-trees (using the ANN library [10]). The codes are written in C++ and run on a 2.26GHz Pentium-III machine. As the "true" segmentation of an image is subjective, so only a visual comparison is intended here.

Table 1: Total wall time (in seconds) on various segmentation tasks, and the number of components in $g(\mathbf{x})$.

| image | data size | mean shift | | our method | |
| | | standard | kd-tree | # components | time consumption |
|---|---|---|---|---|---|
| squirrel | 60,192 (209×288) | 1215.8 | 11.94 | 81 | 0.18 |
| hand | 73,386 (243×302) | 1679.7 | 12.92 | 120 | 0.35 |
| house | 48,960 (192×255) | 1284.5 | 5.16 | 159 | 0.22 |
| lake | 262,144 (512×512) | 3343.0 | 85.65 | 440 | 3.67 |

Segmentation results are shown in Figures 4. The rows, from top to bottom, are: the original image, segmentation results by standard mean shift and our approach. We can see that our results are closer to those by the standard mean shift (applied on the original density estimator), with the number of components (Table 1) dramatically smaller than the data size $n$. This demonstrates the success of our approximation scheme in maintaining the structure of the data distribution using highly compact models. Our algorithm is also much faster than the standard mean shift and its fast version using kd-trees. The reason is that kd-trees only facilitates range searching but does not reduce the expensive computations associated with the large number of kernels.

## 4   Conclusion

Finite mixture is a powerful model in many statistical learning problems. However, the large model size can be a major hindrance in many applications. In this paper, we reduce the model complexity by first grouping the components into compact clusters, and then perform local function approximation that minimizes an upper bound of the approximation error. Our algorithm has low complexity, and demonstrates more reliable performance compared with methods using KL-based distances.

[1] http://www.caip.rutgers.edu/~comanici/MSPAMI/msPamiResults.html

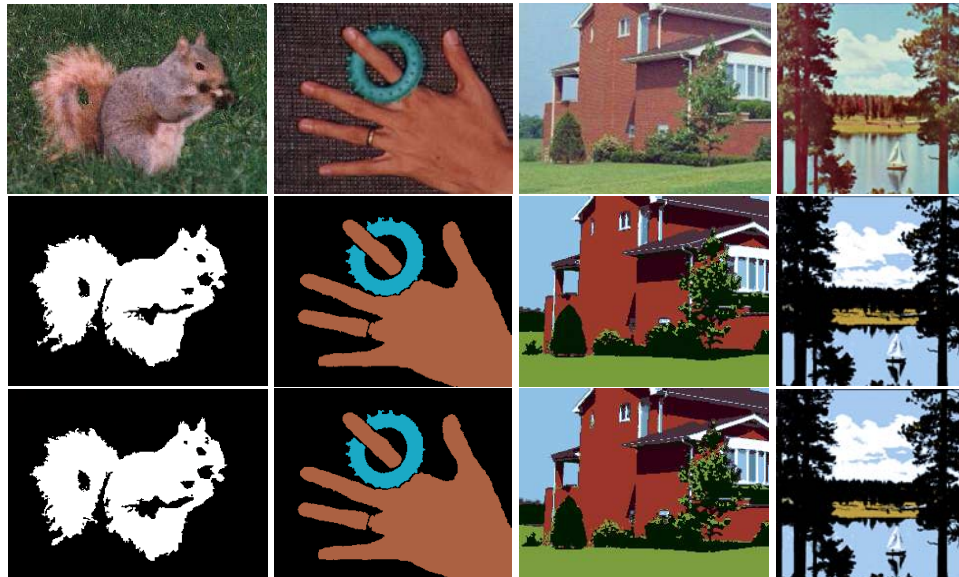

Figure 4: Image segmentation by standard mean shift (2nd row), and ours (bottom).

## References

[1] D. Comaniciu and P. Meer. Mean shift: A robust approach toward feature space analysis. *IEEE Transactions on Pattern Analysis and Machine Intelligence*, 24(5):603–619, 2002.

[2] T. Feder and D. Greene. Optimal algorithms for approximate clustering. In *Proceedings of ACM Symposium on Theory of Computing*, pages 434–444, 1988.

[3] K. Fukunaga and L. Hostetler. The estimation of the gradient of a density function, with applications in pattern recognition. *IEEE Transactions on Information Theory*, 21:32–40, 1975.

[4] A. Gersho and R.M. Gray. *Vector Quantization and Signal Compression*. Kluwer Academic Press, Boston, 1992.

[5] J. Goldberger and S. Roweis. Hierarchical clustering of a mixture model. In *Advances in Neural Information Processing Systems 17*, pages 505–512. 2005.

[6] B. Han, D. Comaniciu, Y. Zhu, and L. Davis. Incremental density approximation and kernel-based Bayesian filtering for object tracking. In *Proceedings of the International Conference on Computer Vision and Pattern Recognition*, pages 638–644, 2004.

[7] A.J. Izenman. Recent developments in nonparametric density estimation. *Journal of the American Statistical Association*, 86(413):205–224, 1991.

[8] T. Kanungo, D.M. Mount, N.S. Netanyahu, C.D. Piatko, R. Silverman, and A.Y. Wu. An efficient $k$-means clustering algorithm: Analysis and implementation. *IEEE Transactions on Pattern Analysis and Machine Intelligence*, 24(7):881–892, 2002.

[9] A.W. Moore. Very fast EM-based mixture model clustering using multiresolution kd-trees. In *Advances in Neural Information Processing Systems 11*, pages 543–549, 1998.

[10] D.M. Mount and S. Arya. ANN: A library for approximate nearest neighbor searching. In *Proceedings of Center for Geometric Computing Second Annual Fall Workshop Computational Geometry (available from* http://www.cs.umd.edu/~mount/ANN*)*, 1997.

[11] E. Parzen. On estimation of a probability density function and mode. *Annals of Mathematical Statistics*, 33:1065–1075, 1962.

[12] K. Popat and R.W. Picard. Cluster-based probability model and its application to image and texture processing. *IEEE Transactions on Image Processing*, 6(2):268–284, 1997.

[13] E.B. Sudderth, A. Torralba, W.T. Freeman, and A.S. Willsky. Describing visual scenes using transformed Dirichlet processes. In *Advances in Neural Information Processing Systems 19*, 2006.

[14] P. Vincent and Y. Bengio. Manifold Parzen windows. In *Advances in Neural Information Processing Systems 15*, 2003.
